# TD$_\gamma$: Re-evaluating Complex Backups in Temporal Difference Learning

**George Konidaris**[*†]
MIT CSAIL[†]
Cambridge MA 02139
gdk@csail.mit.edu

**Scott Niekum**[*‡]
University of Massachusetts Amherst[‡]
Amherst MA 01003
{sniekum,pthomas}@cs.umass.edu

**Philip S. Thomas**[*‡]

## Abstract

We show that the $\lambda$-return target used in the TD($\lambda$) family of algorithms is the maximum likelihood estimator for a specific model of how the variance of an $n$-step return estimate increases with $n$. We introduce the $\gamma$-return estimator, an alternative target based on a more accurate model of variance, which defines the TD$_\gamma$ family of complex-backup temporal difference learning algorithms. We derive TD$_\gamma$, the $\gamma$-return equivalent of the original TD($\lambda$) algorithm, which eliminates the $\lambda$ parameter but can only perform updates at the end of an episode and requires time and space proportional to the episode length. We then derive a second algorithm, TD$_\gamma$($C$), with a capacity parameter $C$. TD$_\gamma$($C$) requires $C$ times more time and memory than TD($\lambda$) and is incremental and online. We show that TD$_\gamma$ outperforms TD($\lambda$) for any setting of $\lambda$ on 4 out of 5 benchmark domains, and that TD$_\gamma$($C$) performs as well as or better than TD$_\gamma$ for intermediate settings of $C$.

## 1 Introduction

Most reinforcement learning [1] algorithms are value-function based—learning is performed by estimating the expected return (discounted sum of rewards) obtained by following the current policy from each state, and then updating the policy based on the resulting so-called *value function*. Efficient value function learning algorithms are crucial to this process and have been the focus of a great deal of reinforcement learning research.

The most successful and widely-used family of value function algorithms is the TD($\lambda$) family [2]. The TD($\lambda$) family forms an estimate of return, called the $\lambda$-return, that blends both low variance, bootstrapped and biased temporal-difference estimates of return with high variance, unbiased Monte Carlo estimates of return using a parameter $\lambda$. While several different algorithms exist within the TD($\lambda$) family—the original incremental and online algorithm [2], replacing traces [3], least-squares algorithms [4], algorithms for learning state-action value functions [1, 5], and algorithms for adapting $\lambda$ [6], among others—the $\lambda$-return formulation has remained unchanged since its introduction in 1988 [2]. Our goal is to understand the modeling assumptions implicit in the $\lambda$-return formulation and improve them.

We show that the $\lambda$-return estimator is only a maximum-likelihood estimator of return given a specific model of how the variance of an $n$-step return estimate increases with $n$. We propose a more accurate model of that variance increase and use it to obtain an expression for a new return estimator, the $\gamma$-return. This results in the TD$_\gamma$ family of algorithms, of which we derive TD$_\gamma$, the $\gamma$-return version of the original TD($\lambda$) algorithm. TD$_\gamma$ is only suitable for the batch setting where updates occur at the end of the episode and requires time and space proportional to the length of the episode,

---

[*]All three authors are primary authors on this occasion.

but it eliminates the $\lambda$ parameter. We then derive a second algorithm, $\text{TD}_\gamma(C)$, which requires $C$ times more time and memory than $\text{TD}(\lambda)$ and can be used in an incremental and online setting. We show that $\text{TD}_\gamma$ outperforms $\text{TD}(\lambda)$ for any setting of $\lambda$ on 4 out of 5 benchmark domains, and that $\text{TD}_\gamma(C)$ performs as well as or better than $\text{TD}_\gamma$ for intermediate settings of $C$.

## 2 Complex Backups

Estimates of return lie at the heart of value-function based reinforcement learning algorithms: a value function $V^\pi$ (which we denote here as $V$, assuming a fixed policy) estimates return from each state, and the learning process aims to reduce the error between estimated and observed returns. Temporal difference (TD) algorithms use a return estimate obtained by taking a single transition in the MDP and then estimating the remaining return using the value function itself:

$$R_{s_t}^{\text{TD}} = r_t + \gamma V(s_{t+1}), \tag{1}$$

where $R_{s_t}^{\text{TD}}$ is the return estimate from state $s_t$ and $r_t$ is the reward for going from $s_t$ to $s_{t+1}$ via action $a_t$. Monte Carlo algorithms (for episodic tasks) do not use intermediate estimates but instead use the full return sample directly:

$$R_{s_t}^{\text{MC}} = \sum_{i=0}^{L-1} \gamma^i r_{t+i}, \tag{2}$$

for an episode $L$ transitions in length after time $t$. These two types of return estimates can be considered instances of the more general notion of an $n$-step return sample, for $n \geq 1$:

$$R_{s_t}^{(n)} = r_t + \gamma r_{t+1} + \gamma^2 r_{t+2} + \ldots + \gamma^{n-1} r_{t+n-1} + \gamma^n V(s_{t+n}). \tag{3}$$

Here, $n$ transitions are observed from the MDP and the remaining portion of return is estimated using the value function. The important observation here is that *all* $n$-step return estimates can be used simultaneously for learning. The $\text{TD}(\lambda)$ family of algorithms accomplishes this using a parameter $\lambda \in [0, 1]$ to average $n$-step return estimates, according to the following equation:

$$R_{s_t}^\lambda = (1 - \lambda) \sum_{n=0}^{\infty} \lambda^n R_{s_t}^{(n+1)}. \tag{4}$$

Note that for any episodic MDP we always obtain a finite episode length. The usual formulation of an episodic MDP uses absorbing terminal states—states where only zero-reward self-transitions are available. In such cases the $n$-step returns past the end of the episode are all equal, and therefore $\text{TD}(\lambda)$ allocates the weights corresponding to all of those return estimates to the final transition.

$R_{s_t}^\lambda$, known as the $\lambda$-return, is an estimator that blends one-step temporal difference estimates (which are biased, but low variance) at $\lambda = 0$ and Monte Carlo estimates (which are unbiased, but high variance) at $\lambda = 1$. This forms the target for the entire family of $\text{TD}(\lambda)$ algorithms, whose members differ largely in their use of the resulting estimates to update the value function.

What makes this a good way to average the $n$-step returns? Why choose this method over any other? Viewing this as a statistical estimation problem where each $n$-step return is a sample of the true return, under what conditions and for what model is equation (4) a good estimator for return?

The most salient feature of the $n$-step returns is that their variances increase with $n$. Therefore, consider the following model: each $n$-step return estimate $R_{s_t}^{(n)}$ is sampled from a Gaussian distribution centered on the true return, $R_{s_t}$,[1] with variance $k(n)$ that is some increasing function of $n$. If we assume the $n$-step returns are independent,[2] then the likelihood function for return estimate $\hat{R}_{s_t}$ is:

$$\mathcal{L}(\hat{R}_{s_t} | R_{s_t}^{(1)}, \ldots, R_{s_t}^{(n)}; k) = \prod_{n=1}^{L} \mathcal{N}(R_{s_t}^{(n)} | \hat{R}_{s_t}, k(n)). \tag{5}$$

Maximizing the log of this equation obtains the maximum likelihood estimator for $\hat{R}_{s_t}$:

$$\hat{R}_{s_t} = \frac{\sum_{n=1}^{L} k(n)^{-1} R_{s_t}^{(n)}}{\sum_{n=1}^{L} k(n)^{-1}}. \tag{6}$$

Thus, we obtain a weighted sum: each $n$-step return is weighted by the inverse of its variance and the entire sum is normalized so that the weights sum to 1. From here we can see that if we let $L$ go to infinity and set $k(n) = \lambda^{-n}$, $0 \leq \lambda \leq 1$, then we obtain the $\lambda$-return estimator in equation (4), since $\sum_{n=0}^{\infty} \lambda^n = 1/(1-\lambda)$.

Thus, $\lambda$-return (as used in the TD($\lambda$) family of algorithms) is the maximum-likelihood estimator of return under the following assumptions:

1. The $n$-step returns from a given state are independent.

2. The $n$-step returns from a given state are normally distributed with a mean of the true return.

3. The variances of the $n$-step returns from each state increase according to a geometric progression in $n$, with common ratio $\lambda^{-1}$.

All of these assumptions could be improved, but the third is the most interesting. In this view, the variance of an $n$-step sample return increases geometrically with $n$ and $\lambda$ parametrizes the shape of this geometric increase.

## 3 $\gamma$-Return and the TD$_\gamma$ Family of Algorithms

Consider the variance of an $n$-step sample return, $n > 1$:

$$Var\left[R_{s_t}^{(n)}\right] = Var\left[R_{s_t}^{(n-1)} - \gamma^{n-1} V(s_{t+n-1}) + \gamma^{n-1} r_{t+n-1} + \gamma^n V(s_{t+n})\right] \tag{7}$$

$$= Var\left[R_{s_t}^{(n-1)}\right] + \gamma^{2(n-1)} Var\left[V(s_{t+n-1}) - (r_{t+n-1} + \gamma V(s_{t+n}))\right] \tag{8}$$

$$+ 2Cov\left[R_{s_t}^{(n-1)}, -\gamma^{n-1} V(s_{t+n-1}) + \gamma^{n-1} r_{t+n-1} + \gamma^n V(s_{t+n})\right].$$

Examining the last term, we see that:

$$Cov\left[R_{s_t}^{(n-1)}, -\gamma^{n-1} V(s_{t+n-1}) + \gamma^{n-1} r_{t+n-1} + \gamma^n V(s_{t+n})\right] \tag{9}$$

$$= Cov\left[R_{s_t}^{(n-1)}, R_{s_t}^{(n)} - R_{s_t}^{(n-1)}\right] \tag{10}$$

$$= Cov\left[R_{s_t}^{(n-1)}, R_{s_t}^{(n)}\right] - Cov\left[R_{s_t}^{(n-1)}, R_{s_t}^{(n-1)}\right] \tag{11}$$

$$= Cov\left[R_{s_t}^{(n-1)}, R_{s_t}^{(n)}\right] - Var\left[R_{s_t}^{(n-1)}\right]. \tag{12}$$

Since $R_{s_t}^{(n-1)}$ and $R_{s_t}^{(n)}$ are highly correlated—being two successive return samples—we assume that $Cov[R_{s_t}^{(n-1)}, R_{s_t}^{(n)}] \approx Var[R_{s_t}^{(n-1)}]$ (equality holds when $R_{s_t}^{(n)}$ and $R_{s_t}^{(n-1)}$ are perfectly correlated). Thus, equation (12) is approximately zero. Hence, equation (8) becomes:

$$Var\left[R_{s_t}^{(n)}\right] \approx Var\left[R_{s_t}^{(n-1)}\right] + \gamma^{2(n-1)} Var\left[V(s_{t+n-1}) - (r_{t+n-1} + \gamma V(s_{t+n}))\right]. \tag{13}$$

Notice that the final term on the right hand side of equation (13) is the discounted variance of the temporal difference error $n$-steps after the current state. We assume that this variance is roughly the same for all states; let that value be $\kappa$. Since $\kappa$ also approximates the variance of the 1-step return (i.e., $k(1) = \kappa$), we obtain the following simple model of the variance of an $n$-step sample of return:

$$k(n) = \sum_{i=1}^{n} \gamma^{2(i-1)} \kappa. \tag{14}$$

Substituting equation (14) into the general return estimator in equation (6), we obtain:

$$R_{s_t}^{\gamma} = \frac{\kappa^{-1} \sum_{n=1}^{L} (\sum_{i=1}^{n} \gamma^{2(i-1)})^{-1} R_{s_t}^{(n)}}{\kappa^{-1} \sum_{n=1}^{L} (\sum_{i=1}^{n} \gamma^{2(i-1)})^{-1}} = \sum_{n=1}^{L} w(n,L) R_{s_t}^{(n)}, \qquad (15)$$

where

$$w(n,L) = \frac{(\sum_{i=1}^{n} \gamma^{2(i-1)})^{-1}}{\sum_{n=1}^{L} (\sum_{i=1}^{n} \gamma^{2(i-1)})^{-1}} \qquad (16)$$

is the weight associated with the $n$th-step return in a trajectory of length $L$ after time $t$. This estimator has the virtue of being *parameter-free* since the $\kappa$ values cancel. Therefore, we need not estimate $\kappa$— under the assumption of independent, Gaussian $n$-step returns with variances increasing according to equation (13), equation (15) is the maximum likelihood estimator *for any value of $\kappa$*. We call this estimator the $\gamma$-return since it weights the $n$-step returns according to the discount factor.

Figure 1 compares the weightings obtained using $\lambda$-return and $\gamma$-return for a few example trajectory lengths. There are two important qualitative differences. First, the $\lambda$-return weightings spike at the end of the trajectory, whereas the $\gamma$-return weightings do not. This occurs because even though any sample trajectory has finite length, the $\lambda$-return as defined in equation (4) is actually an infinite sum; the remainder of the weight mass is allocated to the Monte Carlo return. This allows the normalizing factor in equation (4) to be a constant, rather than having it depend on the length of the trajectory, as it does in equation (15) for the $\gamma$-return. This significantly complicates the problem of obtaining an incremental algorithm using $\gamma$-return, as we shall see in later sections.

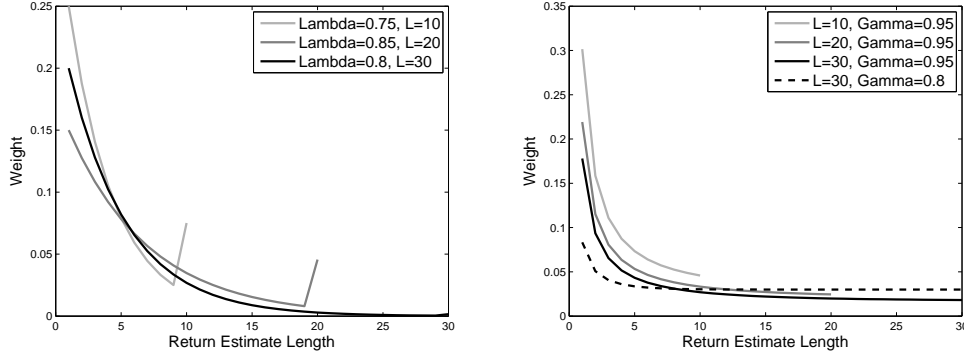

Figure 1: Example weights for trajectories of various lengths for $\lambda$-return (left) and $\gamma$-return (right).

Second, while the $\lambda$-return weightings tend to zero over time, the $\gamma$-return weightings tend toward a constant. This means that very long trajectories will be effectively "cut-off" after some point and have effectively no contribution to the $\lambda$-return, whereas after a certain length in the $\gamma$-return all $n$-step returns have roughly equal weighting. This also complicates the problem of obtaining an incremental algorithm using $\gamma$-return: even if we were to assume infinitely many $n$-step returns past the end of the episode, the normalizing constant would not become finite.

Nevertheless, we can use the $\gamma$-return estimator to obtain an entire family of TD$_\gamma$ learning algorithms; for any TD($\lambda$) algorithm we can derive an equivalent TD$_\gamma$ algorithm. In the following section, we derive TD$_\gamma$, the $\gamma$-return equivalent of the original TD($\lambda$) algorithm.

## 4   TD$_\gamma$

Given a set of $m$ trajectories $T = \{\tau_1, \tau_2, \ldots, \tau_m\}$, where $l_\tau = |\tau|$ denotes the number of $(s_t^\tau, r_t^\tau, s_{t+1}^\tau)$ tuples in the trajectory $\tau$. Using approximator $\hat{V}_\theta$ with parameters $\theta$ to approximate

$V$, the objective function for regression is:

$$E(\theta) = \frac{1}{2} \sum_{\tau \in T} \sum_{t=0}^{l_\tau - 1} \left( R^\gamma_{s^\tau_t} - \hat{V}_\theta(s^\tau_t) \right)^2 \tag{17}$$

$$= \frac{1}{2} \sum_{\tau \in T} \sum_{t=0}^{l_\tau - 1} \left( \sum_{n=1}^{l_\tau - t} w(n, l_\tau - t) R^{(n)}_{s^\tau_t} - \hat{V}_\theta(s^\tau_t) \right)^2. \tag{18}$$

Because $\sum_{n=1}^{l_\tau - t} w(n, l_\tau - t) = 1$, we can write

$$E(\theta) = \frac{1}{2} \sum_{\tau \in T} \sum_{t=0}^{l_\tau - 1} \left( \sum_{n=1}^{l_\tau - t} w(n, l_\tau - t) R^{(n)}_{s^\tau_t} - \sum_{n=1}^{l_\tau - t} w(n, l_\tau - t) \hat{V}_\theta(s^\tau_t) \right)^2 \tag{19}$$

$$= \frac{1}{2} \sum_{\tau \in T} \sum_{t=0}^{l_\tau - 1} \left( \sum_{n=1}^{l_\tau - t} w(n, l_\tau - t) \left[ R^{(n)}_{s^\tau_t} - \hat{V}_\theta(s^\tau_t) \right] \right)^2. \tag{20}$$

Our goal is to minimize $E(\theta)$. One approach is to descend the gradient $\nabla_\theta E(\theta)$, assuming that the $R^{(n)}_{s^\tau_t}$ are noisy samples of $V(s^\tau_t)$ and not a function of $\theta$, as in the derivation of TD($\lambda$) [7]:

$$\Delta\theta = -\alpha \nabla_\theta E(\theta) = -\alpha \sum_{\tau \in T} \sum_{t=0}^{l_\tau - 1} \sum_{n=1}^{l_\tau - t} -w(n, l_\tau - t) \left[ R^{(n)}_{s^\tau_t} - \hat{V}_\theta(s^\tau_t) \right] \nabla_\theta \hat{V}_\theta(s^\tau_t), \tag{21}$$

where $\alpha$ is a learning rate. We can substitute in $n = u - t$ (where $u$ is the current time step, $s_t$ is the state we want to update the value estimate of, and $n$ is the length of the $n$-step return that ends at the current time step) to get:

$$\Delta\theta = -\alpha \sum_{\tau \in T} \sum_{t=0}^{l_\tau - 1} \sum_{u=t+1}^{l_\tau} -w(u - t, l_\tau - t) \left[ R^{(u-t)}_{s^\tau_t} - \hat{V}_\theta(s^\tau_t) \right] \nabla_\theta \hat{V}_\theta(s^\tau_t). \tag{22}$$

Swapping the sums allows us to derive the backward version of TD$_\gamma$:

$$\Delta\theta = -\alpha \sum_{\tau \in T} \sum_{u=1}^{l_\tau} \sum_{t=0}^{u-1} -w(u - t, l_\tau - t) \left[ R^{(u-t)}_{s^\tau_t} - \hat{V}_\theta(s^\tau_t) \right] \nabla_\theta \hat{V}_\theta(s^\tau_t). \tag{23}$$

Expanding and rearranging the terms gives us an algorithm for TD$_\gamma$ when using linear function approximation with weights $\theta$:

$$\Delta\theta = -\alpha \sum_{\tau \in T} \sum_{u=1}^{l_\tau} \sum_{t=0}^{u-1} w(u - t, l_\tau - t) \left[ -\left( \sum_{i=t}^{u-1} \gamma^{i-t} r_{s^\tau_i} \right) - \gamma^{u-t}(\theta \cdot \phi_{s^\tau_u}) + (\theta \cdot \phi_{s^\tau_t}) \right] \phi_{s^\tau_t} \tag{24}$$

$$= -\alpha \sum_{\tau \in T} \sum_{u=1}^{l_\tau} \sum_{t=0}^{u-1} w(u - t, l_\tau - t) \left[ \theta \cdot (\phi_{s^\tau_t} - \gamma^{u-t}\phi_{s^\tau_u}) - \left( \sum_{i=t}^{u-1} \gamma^{i-t} r_{s^\tau_i} \right) \right] \phi_{s^\tau_t} \tag{25}$$

$$= -\alpha \sum_{\tau \in T} \sum_{u=1}^{l_\tau} \sum_{t=0}^{u-1} w(u - t, l_\tau - t) \left[ \theta \cdot \boldsymbol{a} - b \right] \phi_{s^\tau_t}, \tag{26}$$

where $\phi_{s^\tau_t}$ is the feature vector at state $s^\tau_t$, $\boldsymbol{a} = \phi_{s^\tau_t} - \gamma^{u-t}\phi_{s^\tau_u}$, and $b = \sum_{i=t}^{u-1} \gamma^{i-t} r_{s^\tau_i}$. This leads to TD$_\gamma$ for episodic tasks (given in Algorithm 1), which eliminates the eligibility trace parameter $\lambda$. For episode length $l_\tau$ and feature vector size $F$, the algorithm can be implemented with time complexity of $O(l_\tau F)$ per step and space complexity $O(l_\tau F)$. Unfortunately, implementing this backward TD$_\gamma$ incrementally is problematic: $l_\tau$ is not known until the end of the trajectory is reached, and without

---

**Algorithm 1** TD$_\gamma$

---

**Given:** A discount factor, $\gamma$

1: $\boldsymbol{\theta} \leftarrow \vec{0}$
2: **for** each trajectory $\tau \in T$ **do**
3:     Store $\boldsymbol{\phi}_0$ in memory
4:     **for** $u = 1$ to $l_\tau$ **do**
5:         Store $\boldsymbol{\phi}_u$ and $r_{u-1}$ in memory
6:         $\boldsymbol{\delta} \leftarrow \vec{0}$
7:         **for** $t = 0$ to $u - 1$ **do**
8:             $\boldsymbol{a} \leftarrow \boldsymbol{\phi}_t - \gamma^{u-t}\boldsymbol{\phi}_u$
9:             $b \leftarrow \sum\limits_{i=t}^{u-1} \gamma^{i-t} r_i$
10:           $\boldsymbol{\delta} \leftarrow \boldsymbol{\delta} + w(u-t, l_\tau - t)[\boldsymbol{\theta} \cdot \boldsymbol{a} - b]\boldsymbol{\phi}_t$
11:         **end for**
12:         $\boldsymbol{\theta} \leftarrow \boldsymbol{\theta} - \alpha\boldsymbol{\delta}$
13:     **end for**
14:     Discard all $\phi$ and $r$ from memory
15: **end for**

---

it, the normalizing constant of the weights used in the updates cannot be computed. Thus, Algorithm 1 can only be used for batch updates where each episode's trajectory data is stored until an update is performed at the end of an episode; this is often undesirable, and in continuing tasks, impossible.

TD($\lambda$) is able to achieve $O(F)$ time and space for two reasons. First, the weight normalization is a constant and does not depend on the length of the episode. Second, the operation that must be performed on each trace is the same—a multiplication by $\lambda$. Thus, TD($\lambda$) need only store the sum of the return estimates from each state, rather than having to store each individually.

One approach to deriving an incremental algorithm is to use only the first $C$ $n$-step returns from each state, similar to truncated temporal differences [8]. This eliminates the first barrier: weight normalization no longer relies on the episode length, except for the final $C - 1$ states, which can be corrected for after the episode ends. This approach has time complexity $O(CF)$ and space complexity $O(CF)$—and is therefore $C$ times more expensive than TD($\lambda$)—and replaces $\lambda$ with the more intuitive parameter $C$ rather than eliminating it, but it affords the incremental TD$_\gamma(C)$ algorithm given in Algorithm 2. Note that setting $C = 1$ obtains TD(0), and $C \geq l_\tau$ obtains TD$_\gamma$.

## 5 Empirical Comparisons

Figure 2 shows empirical comparisons of TD($\lambda$) (for various values of $\lambda$), TD$_\gamma$ and TD$_\gamma(C)$ for 5 common benchmarks. The first is a $10 \times 10$ discrete gridworld with the goal in the bottom right, deterministic movement in the four cardinal directions, a terminal reward of $+10$ and $-0.5$ for all other transitions, and $\gamma = 1.0$. For the gridworld, the agent selected one of the optimal actions with probability 0.4, and each of the other actions with probability 0.2. The second domain is the 5-state discrete chain domain [1] with random transitions to the left and right, and $\gamma = 0.95$. The third domain is the pendulum swing-up task [7] with a reward of 1.0 for entering a terminal state (the pendulum is nearly vertical) and zero elsewhere, and $\gamma = 0.95$. The optimal action was selected with probability 0.5, with a random action selected otherwise. The fourth domain is mountain car [1] with $\gamma = 0.99$, and using actions from a hand-coded policy with probability 0.75, and random actions otherwise. The fifth and final domain is the acrobot [1] with a terminal reward of 10 and $-0.1$ elsewhere. A random policy was used with $\gamma = 0.95$. In all cases the start state was selected uniformly from the set of nonterminal states. A 5th order Fourier basis [9] was used as the function approximator for the 3 continuous domains. We used 10, 5, 10, 3, and 10 trajectories, respectively.

TD$_\gamma$ outperforms TD($\lambda$) for all settings of $\lambda$ in 4 out of the 5 domains. In the chain domain TD$_\gamma$ performs better than most settings of $\lambda$ but slightly worse than the optimal setting. An interesting and somewhat unexpected result is that TD$_\gamma(C)$ with a relatively low setting of $C$ does at least as well as—or in some cases better than—TD$_\gamma$. This could occur because the $n$-step returns become very

**Algorithm 2** $\text{TD}_\gamma(C)$

---

**Given:** A discount factor, $\gamma$
            A cut-off length, $C$

 1: $\boldsymbol{\theta} \leftarrow \vec{0}$
 2: **for** each trajectory $\tau \in T$ **do**
 3:      Store $\boldsymbol{\phi}_0$ in memory
 4:      **for** $u = 1$ to $l_\tau$ **do**
 5:         If $u > C$, discard $\boldsymbol{\phi}_{u-C-1}$, $\boldsymbol{\theta}_{u-C-1}$, and $\boldsymbol{r}_{u-C-1}$ from memory
 6:         $\boldsymbol{\theta}_{u-1} \leftarrow \boldsymbol{\theta}$
 7:         Store $\boldsymbol{\phi}_u$, $\boldsymbol{\theta}_{u-1}$, and $r_{u-1}$ in memory
 8:         $\boldsymbol{\delta} \leftarrow \vec{0}$
 9:         $m = \max(0, u - C)$
10:         **for** $t = m$ to $u - 1$ **do**
11:            $\boldsymbol{a} \leftarrow \boldsymbol{\phi}_t - \gamma^{u-t} \boldsymbol{\phi}_u$
12:            $b \leftarrow \sum\limits_{i=t}^{u-1} \gamma^{i-t} r_i$
13:            $\boldsymbol{\delta} \leftarrow \boldsymbol{\delta} + w(u-t, C)[\boldsymbol{\theta} \cdot \boldsymbol{a} - b]\boldsymbol{\phi}_t$
14:         **end for**
15:         $\boldsymbol{\theta} \leftarrow \boldsymbol{\theta} - \alpha \boldsymbol{\delta}$
16:      **end for**

17:      $m = \min(l_\tau, C)$
18:      $\boldsymbol{\theta} \leftarrow \boldsymbol{\theta}_{l_\tau - m}$
19:      **for** $\hat{u} = l_\tau - m$ to $l_\tau$ **do**
20:         $\boldsymbol{\delta} \leftarrow \vec{0}$
21:         **for** $t = m$ to $\hat{u} - 1$ **do**
22:            $\boldsymbol{a} \leftarrow \boldsymbol{\phi}_t - \gamma^{\hat{u}-t} \boldsymbol{\phi}_{\hat{u}}$
23:            $b \leftarrow \sum\limits_{i=t}^{\hat{u}-1} \gamma^{i-t} r_i$
24:            $\boldsymbol{\delta} \leftarrow \boldsymbol{\delta} + w(\hat{u}-t, m-t)[\boldsymbol{\theta} \cdot \boldsymbol{a} - b]\boldsymbol{\phi}_t$
25:         **end for**
26:         $\boldsymbol{\theta} \leftarrow \boldsymbol{\theta} - \alpha \boldsymbol{\delta}$
27:         Discard $\boldsymbol{\phi}_{\hat{u}}$, $\boldsymbol{\theta}_{\hat{u}-1}$, and $\boldsymbol{r}_{\hat{u}-1}$ from memory
28:      **end for**
29: **end for**

---

similar for large $n$ due to either $\gamma$ discounting diminishing the difference, or to the additional one-step rewards accounting for a very small fraction of the total return. These near-identical estimates will accumulate a large fraction of the weighting (see Figure 1) and come to dominate the $\gamma$-return estimate. This suggests that once the returns start to become almost identical they should not be considered independent samples and should instead be discarded.

## 6 Discussion and Future Work

An immediate goal of future work is finding an automatic way to set $C$. We may be able to calculate bounds on the diminishing differences between $n$-step returns due to $\gamma$, or empirically track the point at which those differences begin to diminish. Another avenue for future research is deriving a version of $\text{TD}_\gamma$ or $\text{TD}_\gamma(C)$ that provably converges for off-policy data with function approximation, most likely using recent insights on gradient-descent based TD algorithms [10]. Thereafter, we aim to develop an algorithm based on $\gamma$-return for control rather than just prediction, for example Sarsa$_\gamma$.

We have shown that the widely used $\lambda$-return formulation is the maximum-likelihood estimator of return given three assumptions (see section 2). The results presented here have shown that reevaluating just one of these assumptions results in more accurate value function approximation algorithms. We expect that re-evaluating all three will prove a fruitful avenue for future research.

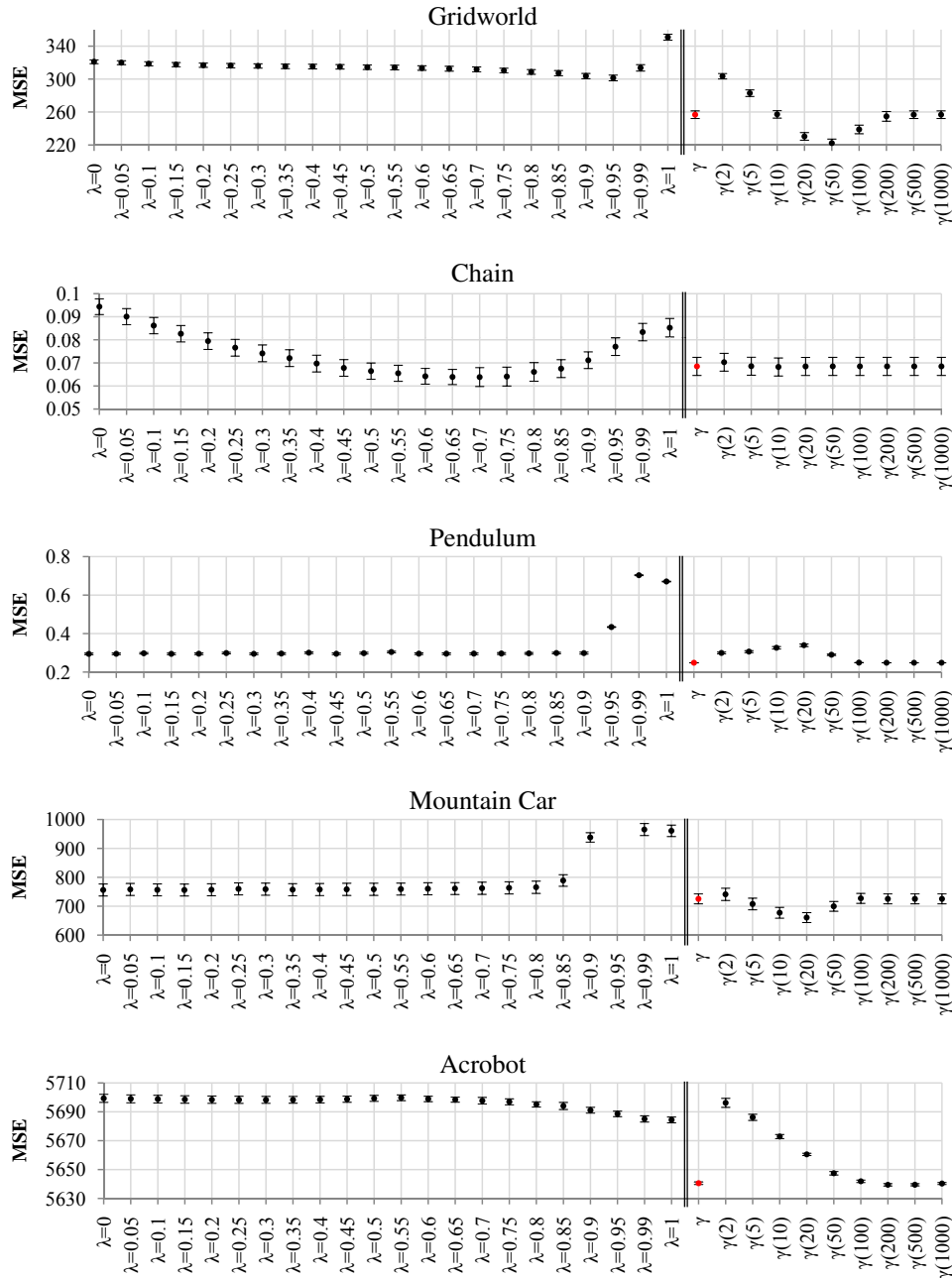

Figure 2: Mean squared error (MSE) over sample trajectories from five benchmark domains for TD($\lambda$) with various settings of $\lambda$, TD$_\gamma$, and TD$_\gamma(C)$, for various settings of $C$. Error bars are standard error over 100 samples. Each result is the minimum MSE (weighted by state visitation frequency) between each algorithm's approximation and the correct value function (obtained using a very large number of Monte Carlo samples), found by searching over $\alpha$ at increments of 0.0001.

**Acknowledgments**

We would like to thank David Silver, Hamid Maei, Gustavo Goretkin, Sridhar Mahadevan and Andy Barto for useful discussions. George Konidaris was supported in part by the AFOSR under grant AOARD-104135 and the Singapore Ministry of Education under a grant to the Singapore-MIT International Design Center. Scott Niekum was supported by the AFOSR under grant FA9550-08-1-0418.

## Footnotes

[1] We should note that this assumption is not quite true: only the Monte Carlo return is unbiased.

[2] Again, this assumption is not true. However, it allows us to obtain a simple, closed-form estimator.

# References

[1] R.S. Sutton and A.G. Barto. *Reinforcement Learning: An Introduction*. MIT Press, Cambridge, MA, 1998.

[2] R.S. Sutton. Learning to predict by the methods of temporal differences. *Machine Learning*, 3(1):9–44, 1988.

[3] S. Singh and R.S. Sutton. Reinforcement learning with replacing eligibility traces. *Machine Learning*, 22:123–158, 1996.

[4] J.A. Boyan. Least squares temporal difference learning. In *Proceedings of the 16th International Conference on Machine Learning*, pages 49–56, 1999.

[5] H.R. Maei and R.S. Sutton. GQ($\lambda$): A general gradient algorithm for temporal-difference prediction learning with eligibility traces. In *Proceedings of the Third Conference on Artificial General Intelligence*, 2010.

[6] C. Downey and S. Sanner. Temporal difference Bayesian model averaging: A Bayesian perspective on adapting lambda. In *Proceedings of the 27th International Conference on Machine Learning*, pages 311–318, 2010.

[7] K. Doya. Reinforcement learning in continuous time and space. *Neural Computation*, 12(1):219–245, 2000.

[8] P. Cichosz. Truncating temporal differences: On the efficient implementation of TD($\lambda$) for reinforcement learning. *Journal of Artificial Intelligence Research*, 2:287–318, 1995.

[9] G.D. Konidaris, S. Osentoski, and P.S. Thomas. Value function approximation in reinforcement learning using the Fourier basis. In *Proceedings of the Twenty-Fifth Conference on Artificial Intelligence*, pages 380–385, 2011.

[10] R.S. Sutton, H.R. Maei, D. Precup, S. Bhatnagar, D. Silver, Cs. Szepesvari, and E. Wiewiora. Fast gradient-descent methods for temporal-difference learning with linear function approximation. In *Proceedings of the 26th International Conference on Machine Learning*, 2009.

